# Incorporating Invariances in Nonlinear Support Vector Machines

**Olivier Chapelle**
olivier.chapelle@lip6.fr
LIP6, Paris, France
Biowulf Technologies

**Bernhard Schölkopf**
bernhard.schoelkopf@tuebingen.mpg.de
Max-Planck-Institute, Tübingen, Germany
Biowulf Technologies

## Abstract

The choice of an SVM kernel corresponds to the choice of a representation of the data in a feature space and, to improve performance, it should therefore incorporate prior knowledge such as known transformation invariances. We propose a technique which extends earlier work and aims at incorporating invariances in nonlinear kernels. We show on a digit recognition task that the proposed approach is superior to the Virtual Support Vector method, which previously had been the method of choice.

## 1 Introduction

In some classification tasks, an a priori knowledge is known about the invariances related to the task. For instance, in image classification, we know that the label of a given image should not change after a small translation or rotation.

More generally, we assume we know a local transformation $\mathcal{L}_t$ depending on a parameter $t$ (for instance, a vertical translation of $t$ pixels) such that any point $\mathbf{x}$ should be considered equivalent to $\mathcal{L}_t\mathbf{x}$, the transformed point. Ideally, the output of the learned function should be constant when its inputs are transformed by the desired invariance.

It has been shown [1] that one can not find a non-trivial kernel which is globally invariant. For this reason, we consider here local invariances and for this purpose we associate at each training point $\mathbf{x}_i$ a *tangent vector* $d\mathbf{x}_i$,

$$d\mathbf{x}_i = \lim_{t \to 0} \frac{1}{t}\left(\mathcal{L}_t\mathbf{x}_i - \mathbf{x}_i\right) = \left.\frac{\partial}{\partial t}\right|_{t=0} \mathcal{L}_t\mathbf{x}_i$$

In practice $d\mathbf{x}_i$ can be either computed by finite difference or by differentiation. Note that generally one can consider more than one invariance transformation.

A common way of introducing invariances in a learning system is to add the perturbed examples $\mathcal{L}_t\mathbf{x}_i$ in the training set [7]. Those points are often called *virtual examples*. In the SVM framework, when applied only to the SVs, it leads to the *Virtual Support Vector* (VSV) method [10]. An alternative to this is to modify directly the cost function in order to take into account the tangent vectors. This

has been successfully applied to neural networks [13] and linear Support Vector Machines [11]. The aim of the present work is to extend these methods to the case of nonlinear SVMs which will be achieved mainly by using the kernel PCA trick [12].

The paper is organized as follows. After introducing the basics of Support Vector Machines in section 2, we recall the method proposed in [11] to train invariant linear SVMs (section 3). In section 4, we show how to extend it to the nonlinear case and finally experimental results are provided in section 5.

## 2 Support Vector Learning

We introduce some standard notations for SVMs; for a complete description, see [15]. Let $\{(\mathbf{x}_i, y_i)\}_{1 \leq i \leq n}$ be a set of training examples, $\mathbf{x}_i \in \mathbb{R}^d$, belonging to classes labeled by $y_i \in \{-1, 1\}$. In kernel methods, we map these vectors into a feature space using a kernel function $K(\mathbf{x}_i, \mathbf{x}_j)$ that defines an inner product in this feature space. The decision function given by an SVM is the maximal margin hyperplane in this space,

$$g(\mathbf{x}) = \text{sign}(f(\mathbf{x})), \text{ where } f(\mathbf{x}) = \left( \sum_{i=1}^{n} \alpha_i^0 y_i K(\mathbf{x}_i, \mathbf{x}) + b \right). \tag{1}$$

The coefficients $\alpha_i^0$ are obtained by maximizing the functional

$$W(\boldsymbol{\alpha}) = \sum_{i=1}^{n} \alpha_i - \frac{1}{2} \sum_{i,j=1}^{n} \alpha_i \alpha_j y_i y_j K(\mathbf{x}_i, \mathbf{x}_j) \tag{2}$$

under the constraints $\sum_{i=1}^{n} \alpha_i y_i = 0$ and $\alpha_i \geq 0$.

This formulation of the SVM optimization problem is called the *hard margin* formulation since no training errors are allowed. In the rest of the paper, we will not consider the so called *soft-margin* SVM algorithm [4], where training errors are allowed.

## 3 Invariances for Linear SVMs

For linear SVMs, one wants to find a hyperplane whose normal vector $\mathbf{w}$ is as orthogonal as possible to the tangent vectors. This can be easily understood from the equality

$$f(\mathbf{x}_i + d\mathbf{x}_i) - f(\mathbf{x}_i) = \mathbf{w} \cdot d\mathbf{x}_i.$$

For this purpose, it has been suggested [11] to minimize the functional

$$(1 - \gamma)\mathbf{w}^2 + \gamma \sum_{i=1}^{n} (\mathbf{w} \cdot d\mathbf{x}_i)^2 \tag{3}$$

subject to the constraints $y_i(\mathbf{w} \cdot \mathbf{x}_i + b) \geq 1$. The parameter $\gamma$ trades off between normal SVM training ($\gamma = 0$) and full enforcement of the orthogonality between the hyperplane and the invariance directions ($\gamma \to 1$).

Let us introduce

$$C_\gamma = \left( (1 - \gamma)I + \gamma \sum_i d\mathbf{x}_i d\mathbf{x}_i^\top \right)^{1/2}, \tag{4}$$

the square root of the regularized covariance matrix of the tangent vectors.

It was shown in [11] that training a linear invariant SVM, i.e. minimizing (3), is equivalent to a standard SVM training after the following linear transformation of the input space

$$\mathbf{x} \to C_\gamma^{-1}\mathbf{x}.$$

This method led to significant improvements in linear SVMs, and to small improvements when used as a linear preprocessing step in *non*linear SVMs. The latter, however, was a hybrid system with unclear theoretical foundations. In the next section we show how to deal with the nonlinear case in a principled way.

## 4  Extension to the nonlinear case

In the nonlinear case, the data are first mapped into a high-dimensional feature space where a linear decision boundary is computed. To extend directly the previous analysis to the nonlinear case, one would need to compute the matrix $C_\gamma$ in feature space,

$$C_\gamma = \left( (1-\gamma)I + \gamma \sum_i d\Phi(\mathbf{x}_i) d\Phi(\mathbf{x}_i)^\top \right)^{1/2} \tag{5}$$

and the new kernel function

$$\tilde{K}(\mathbf{x},\mathbf{y}) = C_\gamma^{-1}\Phi(\mathbf{x}) \cdot C_\gamma^{-1}\Phi(\mathbf{y}) = \Phi(\mathbf{x})^\top C_\gamma^{-2}\Phi(\mathbf{y}) \tag{6}$$

However, due to the high dimension of the feature space, it is impossible to do it directly. We propose two different ways for overcoming this difficulty.

### 4.1  Decomposition of the tangent Gram matrix

In order to be able to compute the new kernel (6), we propose to diagonalize the matrix $C_\gamma$ (eq 5) using a similar approach as the kernel PCA trick [12]. In that article, they showed how it was possible to diagonalize the feature space covariance matrix by computing the eigendecomposition of the Gram matrix of those points. Presently, instead of having a set of training points $\{\Phi(\mathbf{x}_i)\}$, we have a set of tangent vectors $\{d\Phi(\mathbf{x}_i)\}$ and a tangent covariance matrix (the right term of the sum in (5))

Let us introduce the Gram matrix $K^t$ of the tangent vectors:

$$\begin{aligned}
K^t_{ij} &= d\Phi(\mathbf{x}_i) \cdot d\Phi(\mathbf{x}_j) \\
&= K(\mathbf{x}_i+d\mathbf{x}_i, \mathbf{x}_j+d\mathbf{x}_j) - K(\mathbf{x}_i+d\mathbf{x}_i, \mathbf{x}_j) - K(\mathbf{x}_i, \mathbf{x}_j+d\mathbf{x}_j) + K(\mathbf{x}_i, \mathbf{x}_j) \quad (7) \\
&= d\mathbf{x}_i^\top \frac{\partial^2 K(\mathbf{x}_i,\mathbf{x}_j)}{\partial \mathbf{x}_i \partial \mathbf{x}_j} d\mathbf{x}_j \tag{8}
\end{aligned}$$

This matrix $K^t$ can be computed either by finite differences (equation 7) or with the analytical derivative expression given by equation (8). Note that for a linear kernel, $K(\mathbf{x},\mathbf{y}) = \mathbf{x}^\top\mathbf{y}$, and (8) reads $K^t_{ij} = d\mathbf{x}_i^\top d\mathbf{x}_j$, which is a standard dot product between the tangent vectors.

Writing the eigendecomposition of $K^t$ as $K^t = U\Lambda U^\top$, and using the kernel PCA tools [12], one can show after some algebra (details in [2]) that the new kernel matrix reads

$$\begin{aligned}
\tilde{K}(\mathbf{x},\mathbf{y}) &= \frac{1}{1-\gamma}K(\mathbf{x},\mathbf{y}) + \sum_{p=1}^n \frac{1}{\lambda_p}\left( \frac{1}{\gamma\lambda_p + 1 - \gamma} - \frac{1}{1-\gamma} \right) \\
&\left( \sum_{i=1}^n U_{ip}\, d\mathbf{x}_i^\top \frac{\partial K(\mathbf{x}_i,\mathbf{x})}{\partial \mathbf{x}_i} \right) \left( \sum_{i=1}^n U_{ip}\, d\mathbf{x}_i^\top \frac{\partial K(\mathbf{x}_i,\mathbf{y})}{\partial \mathbf{x}_i} \right)
\end{aligned}$$

## 4.2 The kernel PCA map

A drawback of the previous approach appears when one wants to deal with multiple invariances (i.e. more than one tangent vector per training point). Indeed, it requires to diagonalize the matrix $K^t$ (cf eq 7), whose size is equal to the number of different tangent vectors. For this reason, we propose an alternative method. The idea is to use directly the so-called *kernel PCA map*, first introduced in [12] and extended in [14].

This map is based on the fact that even in a high dimensional feature space $\mathcal{H}$, a training set $\{\mathbf{x}_1, \ldots, \mathbf{x}_n\}$ of size $n$ when mapped to this feature space spans a subspace $E \subset \mathcal{H}$ whose dimension is at most $n$. More precisely, if $(\mathbf{v}_1, \ldots, \mathbf{v}_n) \in E^n$ is an orthonormal basis of $E$ with each $\mathbf{v}_i$ being a principal axis of $\{\Phi(\mathbf{x}_1), \ldots, \Phi(\mathbf{x}_n)\}$, the kernel PCA map $\psi : \mathcal{X} \to \mathbb{R}^n$ is defined coordinatewise as

$$\psi_p(\mathbf{x}) = \Phi(\mathbf{x}) \cdot \mathbf{v}_p, \quad 1 \leq p \leq n.$$

Each principal direction has a linear expansion on the training points $\{\Phi(\mathbf{x}_i)\}$ and the coefficients of this expansion are obtained using kernel PCA [12]. Writing the eigendecompostion of $K$ as $K = U\Lambda U^\top$, with $U$ an orthonormal matrix and $\Lambda$ a diagonal one, it turns out that the the kernel PCA map reads

$$\psi(\mathbf{x}) = \Lambda^{-1/2} U^\top \mathbf{k}(\mathbf{x}), \tag{9}$$

where $\mathbf{k}(\mathbf{x}) = (K(\mathbf{x}, \mathbf{x}_1), \ldots, K(\mathbf{x}, \mathbf{x}_n))^\top$.

Note that by definition, for all $i$ and $j$, $\Phi(\mathbf{x}_i)$ and $\Phi(\mathbf{x}_j)$ lie in $E$ and thus $K(\mathbf{x}_i, \mathbf{x}_j) = \Phi(\mathbf{x}_i) \cdot \Phi(\mathbf{x}_j) = \psi(\mathbf{x}_i) \cdot \psi(\mathbf{x}_j)$. This reflects the fact that if we retain all principal components, kernel PCA is just a basis transform in $E$, leaving the dot product of training points invariant.

As a consequence, training a nonlinear SVM on $\{\mathbf{x}_1, \ldots, \mathbf{x}_n\}$ is equivalent to training a linear SVM on $\{\psi(\mathbf{x}_1), \ldots, \psi(\mathbf{x}_n)\}$ and thus, thanks to the nonlinear mapping $\psi$, we can work directly in the linear space $E$ and use exactly the technique described for invariant *linear* SVMs (section 3). However the invariance directions $d\Phi(\mathbf{x}_i)$ do not necessarily belong to $E$. By projecting them onto $E$, some information might be lost. The hope is that this approximation will give a similar decision function to the exact one obtained in section 4.1.

Finally, the proposed algorithm consists in training an invariant linear SVM as described in section 3 with training set $\{\psi(\mathbf{x}_1), \ldots, \psi(\mathbf{x}_n)\}$ and with invariance directions $\{d\psi(\mathbf{x}_1), \ldots, d\psi(\mathbf{x}_n)\}$, where $d\psi(\mathbf{x}_i) = \psi(\mathbf{x}_i + d\mathbf{x}_i) - \psi(\mathbf{x}_i)$, which can be expressed from equation (9) as

$$d\psi(\mathbf{x}_i)_p = \frac{d\mathbf{x}_i^\top}{\sqrt{\lambda_p}} \sum_{j=1}^n U_{jp} \frac{\partial K(\mathbf{x}_i, \mathbf{x}_j)}{\partial \mathbf{x}_i}.$$

## 4.3 Comparisons with the VSV method

One might wonder what is the difference between enforcing an invariance and just adding the virtual examples $\mathcal{L}_t \mathbf{x}_i$ in the training set. Indeed the two approaches are related and some equivalence can be shown [6].

So why not just add virtual examples ? This is the idea of the Virtual Support Vector (VSV) method [10]. The reason is the following: if a training point $\mathbf{x}_i$ is far from the margin, adding the virtual example $\mathcal{L}_t \mathbf{x}_i$ will not change the decision boundary since neither of the points can become a support vector. Hence adding

virtual examples in the SVM framework enforces invariance *only around the decision boundary*, which, as an aside, is the main reason why the virtual SV method only adds virtual examples generated from points that were support vectors in the earlier iteration.

One might argue that the points which are far from the decision boundary do not provide any information anyway. On the other hand, there is some merit in not only keeping the output label invariant under the transformation $\mathcal{L}_t$, but also the *real-valued output*. This can be justified by seeing the distance of a given point to the margin as an indication of its class-conditional probability [8]. It appears reasonable that an invariance transformation should not affect this probability too much.

# 5 Experiments

In our experiments, we compared a standard SVM with several methods taking into account invariances: standard SVM with virtual examples (cf. the VSV method [10]) [VSV], invariant SVM as described in section 4.1 [ISVM] and invariant hyperplane in kernel PCA coordinates as described in section 4.2 [ $\text{IH}_{\text{KPCA}}$ ].

The hybrid method described in [11] (see end of section 3) did not perform better than the VSV method and is not included in our experiments for this reason.

Note that in the following experiments, each tangent vector $d\Phi(\mathbf{x}_i)$ has been normalized by the average length $\sqrt{\sum \|d\Phi(\mathbf{x}_i)\|^2/n}$ in order to be scale independent.

## 5.1 Toy problem

The toy problem we considered is the following: the training data has been generated uniformly from $[-1,1]^2$. The true decision boundary is a circle centered at the origin: $f(\mathbf{x}) = \text{sign}(\mathbf{x}^2 - 0.7)$.

The a priori knowledge we want to encode in this toy problem is *local invariance under rotations*. Therefore, the output of the decision function on a given training point $\mathbf{x}_i$ and on its image $R(\mathbf{x}_i, \varepsilon)$ obtained by a small rotation should be as similar as possible. To each training point, we associate a tangent vector $d\mathbf{x}_i$ which is actually orthogonal to $\mathbf{x}_i$.

A training set of 30 points was generated and the experiments were repeated 100 times. A Gaussian kernel $K(\mathbf{x}, \mathbf{y}) = \exp\left(-\frac{\|\mathbf{x}-\mathbf{y}\|^2}{2\sigma^2}\right)$ was chosen.

The results are summarized in figure 1. Adding virtual examples (VSV method) is already very useful since it made the test error decrease from 6.25% to 3.87% (with the best choice of $\sigma$). But the use of ISVM or $\text{IH}_{\text{KPCA}}$ yields even better performance. On this toy problem, the more the invariances are enforced ($\gamma \rightarrow 1$), the better the performances are (see right side of figure 1), reaching a test error of 1.11%.

When comparing $\log \sigma = 1.4$ and $\log \sigma = 0$ (right side of of figure 1), one notices that the decrease in the test error does not have the same speed. This is actually the dual of the phenomenon observed on the left side of this figure : for a same value of gamma, the test error tends to increase, when $\sigma$ is larger. This analysis suggests that $\gamma$ needs to be adapted in function of $\sigma$. This can be done automatically by the gradient descent technique described in [3].

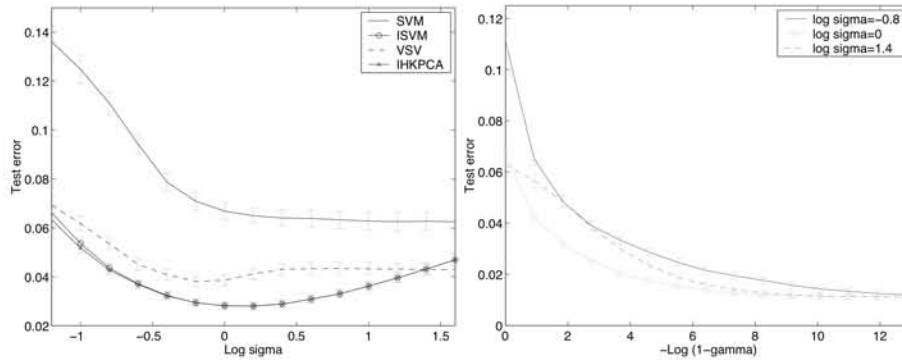

Figure 1: Left: test error for different learning algorithms plotted against the width of a RBF kernel and $\gamma$ fixed to 0.9. Right: test error of $\mathrm{IH_{KPCA}}$ across $\gamma$ and for different values of $\sigma$. The test errors are averaged over the 100 splits and the error bars correspond to the standard deviation of the means.

## 5.2   Handwritten digit recognition

As a real world experiment, we tried to incorporate invariances for a handwritten digit recognition task. The USPS dataset have been used extensively in the past for this purpose, especially in the SVM community. It consists of 7291 training and 2007 test examples.

According to [9], the best performance has been obtained for a polynomial kernel of degree 3, and all the results described in this section were performed using this kernel. The local transformations we considered are translations (horizontal and vertical). All the tangent vectors have been computed by a finite difference between the original digit and its 1-pixel translated.

We split the training set into 23 subsets of 317 training examples after a random permutation of the training and test set. Also we concentrated on a binary classification problem, namely separating digits 0 to 4 against 5 to 9. The gain in performance should also be valid for the multiclass case.

Figure 2 compares ISVM, $\mathrm{IH_{KPCA}}$ and VSV for different values of $\gamma$. From those figures, it can be seen that the difference between ISVM (the original method) and $\mathrm{IH_{KPCA}}$ (the approximation) is much larger than in the toy example. The difference to the toy example is probably due to the input dimensionality. In 2 dimensions, with an RBF kernel, the 30 examples of the toy problem "almost span" the whole feature space, whereas with 256 dimensions, this is no longer the case.

What is noteworthy in these experiments is that our proposed method is much better than the standard VSV. As explained in section 4.3, the reason for this might be that invariance is enforced around *all* training points and not only around support vectors. Note that what we call VSV here is a standard SVM with a *double size* training set containing the original data points and their translates.

The horizontal invariance yields larger improvements than the vertical one. One of the reason might be that the digits in the USPS database are already centered vertically.

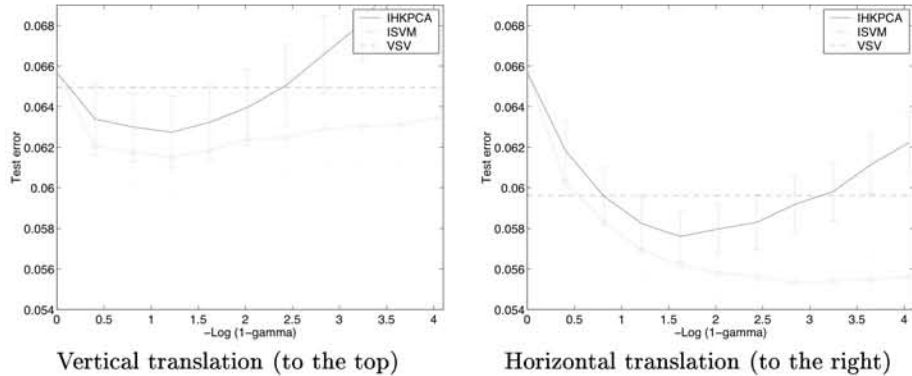

| Vertical translation (to the top) | Horizontal translation (to the right) |

Figure 2: Comparison of ISVM, $IH_{KPCA}$ and VSV on the USPS dataset. The left of the plot ($\gamma = 0$) corresponds to standard SVM whereas the right part of the plot ($\gamma \to 1$) means that a lot of emphasis is put on the enforcement of the constraints. The test errors are averaged over the 23 splits and the error bars correspond to the standard deviation of the means.

## 6   Conclusion

We have extended a method for constructing invariant hyperplanes to the nonlinear case. We have shown results that are superior to the virtual SV method. The latter has recently broken the record on the NIST database which is the "gold standard" of handwritten digit benchmarks [5], therefore it appears promising to also try the new system on that task. For this propose, a large scale version of this method needs to be derived. The first idea we tried is to compute the kernel PCA map using only a subset of the training points. Encouraging results have been obtained on the 10-class USPS database (with the whole training set), but other methods are also currently under study.

## References

[1] C. J. C. Burges. Geometry and invariance in kernel based methods. In B. Schölkopf, C. J. C. Burges, and A. J. Smola, editors, *Advances in Kernel Methods — Support Vector Learning*. MIT Press, 1999.

[2] O. Chapelle and B. Schölkopf. Incorporating invariances in nonlinear Support Vector Machines, 2001. Availabe at: www-connex.lip6.fr/~chapelle.

[3] O. Chapelle, V. Vapnik, O. Bousquet, and S. Mukherjee. Choosing multiple parameters for support vector machines. *Machine Learning*, 46:131–159, 2002.

[4] C. Cortes and V. Vapnik. Support vector networks. *Machine Learning*, 20:273 – 297, 1995.

[5] D. DeCoste and B. Schölkopf. Training invariant support vector machines. *Machine Learning*, 2001. In press.

[6] Todd K. Leen. From data distributions to regularization in invariant learning. In *Nips*, volume 7. The MIT Press, 1995.

[7] P. Niyogi, T. Poggio, and F. Girosi. Incorporating prior information in machine learning by creating virtual examples. *IEEE Proceedings on Intelligent Signal Processing*, 86(11):2196–2209, November 1998.

[8] John Platt. Probabilities for support vector machines. In A. Smola, P. Bartlett, B. Schölkopf, and D. Schuurmans, editors, *Advances in Large Margin Classifiers*. MIT Press, Cambridge, MA, 2000.

[9] B. Schölkopf, C. Burges, and V. Vapnik. Extracting support data for a given task. In U. M. Fayyad and R. Uthurusamy, editors, *First International Conference on Knowledge Discovery & Data Mining*. AAAI Press, 1995.

[10] B. Schölkopf, C. Burges, and V. Vapnik. Incorporating invariances in support vector learning machines. In *Artificial Neural Networks — ICANN'96*, volume 1112, pages 47–52, Berlin, 1996. Springer Lecture Notes in Computer Science.

[11] B. Schölkopf, P. Y. Simard, A. J. Smola, and V. N. Vapnik. Prior knowledge in support vector kernels. In MIT Press, editor, *NIPS*, volume 10, 1998.

[12] B. Schölkopf, A. Smola, and K.-R. Müller. Nonlinear component analysis as a kernel eigenvalue problem. *Neural Computation*, 10:1299–1310, 1998.

[13] P. Simard, Y. LeCun, J. Denker, and B. Victorri. Transformation invariance in pattern recognition, tangent distance and tangent propagation. In G. Orr and K. Muller, editors, *Neural Networks: Tricks of the trade*. Springer, 1998.

[14] K. Tsuda. Support vector classifier with asymmetric kernel function. In M. Verleysen, editor, *Proceedings of ESANN'99*, pages 183–188, 1999.

[15] V. Vapnik. *Statistical Learning Theory*. John Wiley & Sons, 1998.
